# Discriminative Keyword Selection Using Support Vector Machines

**W. M. Campbell, F. S. Richardson**
MIT Lincoln Laboratory
Lexington, MA 02420
*wcampbell,frichard@ll.mit.edu*

## Abstract

Many tasks in speech processing involve classification of long term characteristics of a speech segment such as language, speaker, dialect, or topic. A natural technique for determining these characteristics is to first convert the input speech into a sequence of tokens such as words, phones, etc. From these tokens, we can then look for distinctive sequences, *keywords*, that characterize the speech. In many applications, a set of distinctive keywords may not be known *a priori*. In this case, an automatic method of building up keywords from short context units such as phones is desirable. We propose a method for the construction of keywords based upon Support Vector Machines. We cast the problem of keyword selection as a feature selection problem for $n$-grams of phones. We propose an alternating filter-wrapper method that builds successively longer keywords. Application of this method to language recognition and topic recognition tasks shows that the technique produces interesting and significant qualitative and quantitative results.

## 1 Introduction

A common problem in speech processing is to identify properties of a speech segment such as the language, speaker, topic, or dialect. A typical solution to this problem is to apply a detection paradigm. A set of classifiers is applied to a speech segment to produce a decision. For instance, for language recognition, we might construct detectors for English, French, and Spanish. The maximum scoring detector on a speech segment would be the predicted language.

Two basic categories of systems have been applied to the detection problem. A first approach uses short-term spectral characteristics of the speech and models these with Gaussian mixture models (GMMs) or support vector machines (SVMs) directly producing a decision. Although quite accurate, this type of system produces only a classification decision with no qualitative interpretation. A second approach uses *high level* features of the speech such as phones and words to detect the properties. An advantage of this approach is that, in some instances, we can explain why we made a decision. For example, a particular phone or word sequence might indicate the topic. We adopt this latter approach for our paper.

SVMs have become a common method of extracting high-level properties of sequences of speech tokens [1, 2, 3, 4]. Sequence kernels are constructed by viewing a speech segment as a *document* of tokens. The SVM feature space in this case is a scaling of co-occurrence probabilities of tokens in an utterance. This technique is analogous to methods for applying SVMs to text classification [5].

SVMs have been applied at many linguistic levels of tokens as detectors. Our focus in this paper is at the acoustic phone level. Our goal is to automatically derive long sequences of phones which

we call *keywords* which are characteristic of a given class. Prior work, for example, in language recognition [6], has shown that certain words are a significant predictor of a language. For instance, the presence of the phrase "you know" in a conversational speech segment is a strong indicator of English. A difficulty in using words as the indicator of the language is that we may not have available a speech-to-text (STT) system in all languages of interest. In this case, we'd like to automatically construct keywords that are indicative of the language. Note that a similar problem can occur in other property extraction problems. For instance, in topic recognition, proper names not in our STT system dictionary may be a strong indicator of topic.

Our basic approach is to view keyword construction as a feature selection problem. Keywords are composed of sequences of phones of length $n$, i.e. $n$-grams. We would like to find the set of $n$-grams that best discriminates between classes. Unfortunately, this problem is difficult to solve directly, since the number of unique $n$-grams grows exponentially with increasing $n$. To alleviate this difficultly, we propose a method that starts with lower order $n$-grams and successively builds higher order $n$-grams.

The outline of the paper is as follows. In Section 2.1, we review the basic architecture that we use for phone recognition and how it is applied to the problem. In Section 2.2, we review the application of SVMs to determining properties. Section 3.1 describes a feature selection method for SVMs. Section 3.2 presents our method for constructing long context units of phones to automatically create keywords. We use a novel feature selection approach that attempts to find longer strings that discriminate well between classes. Finally, in Section 4, we show the application of our method to language and topic recognition problems. We show qualitatively that the method produces interesting keywords. Quantitatively, we show that the method produces keywords which are good discriminators between classes.

## 2 Phonotactic Classification

### 2.1 Phone Recognition

The high-level token extraction component of our system is a phone recognizer based upon the Brno University (BUT) design [7]. The basic architecture of this system is a monophone HMM system with a null grammar. Monophones are modeled by three states. This system uses two powerful components to achieve high accuracy. First, split temporal context (STC) features provide contextual cues for modeling monophones. Second, the BUT recognizer extensively uses discriminatively trained feedforward artificial neural networks (ANNs) to model HMM state posterior probabilities.

We developed a phone recognizer for English units using the BUT architecture and automatically generated STT transcripts on the Switchboard 2 Cell corpora [8]. Training data consisted of approximately 10 hours of speech. ANN training was accomplished using the ICSI Quicknet package [9]. The resulting system has 49 monophones including silence.

The BUT recognizer is used along with the HTK HMM toolkit [10] to produce lattices. Lattices encode multiple hypotheses with acoustic likelihoods. From a lattice, a 1-best (Viterbi) output can be produced. Alternatively, we use the lattice to produce expected counts of tokens and $n$-grams of tokens.

Expected counts of $n$-grams can be easily understood as an extension of standard counts. Suppose we have a hypothesized string of tokens, $W = w_1, \cdots, w_n$. Then bigrams are created by grouping two tokens at a time to form, $W_2 = w_1\_w_2, w_2\_w_3, \cdots, w_{n-1}\_w_n$. Higher order $n$-grams are formed from longer juxtapositions of tokens. The count function for a given bigram, $d_i$, is $\text{count}(d_i|W_2)$ is the number of occurrences of $d_i$ in the sequence $W_2$. To extend counts to a lattice, $\mathcal{L}$, we find the expected count over all all possible hypotheses in the lattice,

$$\text{count}(d_i|\mathcal{L}) = E_W[\text{count}(d_i|W)] = \sum_{W \in \mathcal{L}} p(W|\mathcal{L})\,\text{count}(d_i|W). \tag{1}$$

The expected counts can be computed efficiently by a forward-backward algorithm; more details can be found in Section 3.3 and [11].

A useful application of expected counts is to find the probability of an $n$-gram in a lattice. For a lattice, $\mathcal{L}$, the joint probability of an $n$-gram, $d_i$, is

$$p(d_i|\mathcal{L}) = \frac{\text{count}(d_i|\mathcal{L})}{\sum_j \text{count}(d_j|\mathcal{L})} \qquad (2)$$

where the sum in (2) is performed over all *unique* $n$-grams in the utterance.

## 2.2  Discriminative Language Modeling: SVMs

We focus on token-based language recognition with SVMs using the approach from [1, 4]. Similar to [1], a lattice of tokens, $\mathcal{L}$, is modeled using a bag-of-$n$-grams approach. Joint probabilities of the unique $n$-grams, $d_j$, on a per conversation basis are calculated, $p(d_j|\mathcal{L})$, see (2). Then, the probabilities are mapped to a sparse vector with entries

$$D_j p(d_j|W). \qquad (3)$$

The selection of the weighting, $D_j$, in (3) is critical for good performance. A typical choice is of the form

$$D_j = \min\left(C_j, g_j\left(\frac{1}{p(d_j|\text{all})}\right)\right) \qquad (4)$$

where $g_j(\cdot)$ is a function which squashes the dynamic range, and $C_j$ is a constant. The probability $p(d_j|\text{all})$ in (4) is calculated from the observed probability across all classes. The squashing function should monotonically map the interval $[1, \infty)$ to itself to suppress large inverse probabilities. Typical choices for $g_j$ are $g_j(x) = \sqrt{x}$ and $g_j(x) = \log(x) + 1$. In both cases, the squashing function $g_j$ normalizes out the typicality of a feature across all classes. The constant $C_j$ limits the effect of any one feature on the kernel inner product. If we set $C_j = 1$, then this makes $D_j = 1$ for all $j$. For the experiments in this paper, we use $g_j(x) = \sqrt{x}$, which is suited to high frequency token streams.

The general weighting of probabilities is then combined to form a kernel between two lattices, see [1] for more details. For two lattices, $\mathcal{L}_1$ and $\mathcal{L}_2$, the kernel is

$$K(\mathcal{L}_1, \mathcal{L}_2) = \sum_j D_j^2 p(d_j|\mathcal{L}_1) p(d_j|\mathcal{L}_2). \qquad (5)$$

Intuitively, the kernel in (5) says that if the same $n$-grams are present in two sequences and the normalized frequencies are similar there will be a high degree of similarity (a large inner product). If $n$-grams are not present, then this will reduce similarity since one of the probabilities in (5) will be zero. The normalization $D_j$ insures that $n$-grams with large probabilities do not dominate the kernel function. The kernel can alternatively be viewed as a linearization of the log-likelihood ratio [1].

Incorporating the kernel (5) into an SVM system is straightforward. SVM training and scoring require only a method of kernel evaluation between two objects that produces positive definite kernel matrices (the Mercer condition). We use the package SVMTorch [12]. Training is performed with a one-versus-all strategy. For each target class, we group all remaining class data and then train with these two classes.

## 3  Discriminative Keyword Selection

### 3.1  SVM Feature Selection

A first step towards an algorithm for automatic keyword generation using phones is to examine feature selection methods. Ideally, we would like to select over all possible $n$-grams, where $n$ is varying, the most discriminative sequences for determining a property of a speech segment. The number of features in this case is prohibitive, since it grows exponentially with $n$. Therefore, we have to consider alternate methods.

As a first step, we examine feature selection for fixed $n$ and look for keywords with $n$ or less phones. Suppose that we have a set of candidate keywords. Since we are already using an SVM, a natural algorithm for discriminative feature selection in this case is to use a wrapper method [13].

Suppose that the optimized SVM solution is

$$f(X) = \sum_i \alpha_i K(X, X_i) + c \tag{6}$$

and

$$\mathbf{w} = \sum_i \alpha_i \mathbf{b}(X_i) \tag{7}$$

where $\mathbf{b}(X_i)$ is the vector of weighted $n$-gram probabilities in (3). We note that the kernel presented in (5) is linear. Also, the $n$-gram probabilities have been normalized in (3) by their probability across the entire data set. Intuitively, because of this normalization and since $f(X) = \mathbf{w}^t \mathbf{b}(X) + c$, large magnitude entries in $\mathbf{w}$ correspond to significant features.

A confirmation of this intuitive idea is the algorithm of Guyon, et. al. [14]. Guyon proposes an iterative wrapper method for feature selection for SVMs which has these basic steps:

- For a set of features, $\mathcal{S}$, find the SVM solution with model $\mathbf{w}$.

- Rank the features by their corresponding model entries $w_i^2$. Here, $w_i$ is the $i$th entry of $\mathbf{w}$ in (7).

- Eliminate low ranking features using a threshold.

The algorithm may be iterated multiple times.

Guyon's algorithm for feature selection can be used for picking significant $n$-grams as keywords. We can create a kernel which is the sum of kernels as in (5) up to the desired $n$. We then train an SVM and rank $n$-grams according to the magnitude of the entries in the SVM model vector, $\mathbf{w}$.

As an example, we have looked at this feature selection method for a language recognition task with trigrams (to be described in Section 4). Figure 1 provides a motivation for the applicability of Guyon's feature selection method. The figure shows two functions. First, the cumulative density function (CDF) of the SVM model values, $|w_i|$, is shown. The CDF has an S-curve shape; i.e., only a small set of models weights has large magnitudes. The second curve shows the equal error rate (EER) of the task as a function of applying one iteration of the Guyon algorithm and retraining the SVM. EER is defined as the value where the miss and false alarm rates are equal. All features with $|w_i|$ below the value on the x-axis are discarded in the first iteration. From the figure, we see that only a small fraction ($< 5\%$) of the features are needed to obtain good error rates. This interesting result provides motivation that a small subset of keywords are significant to the task.

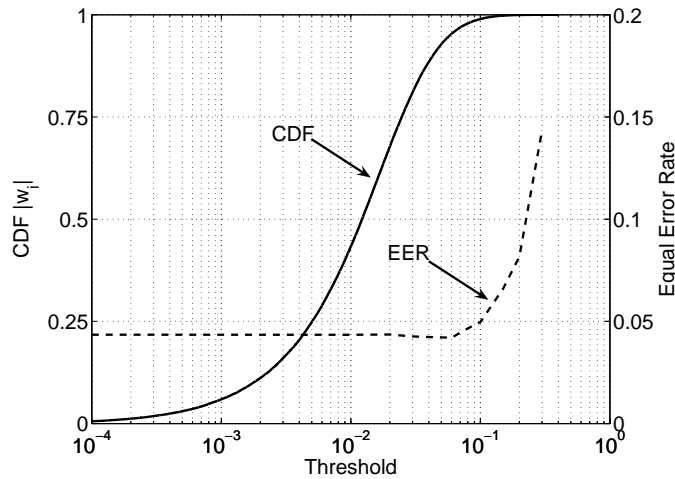

Figure 1: Feature selection for a trigram language recognition task using Guyon's method

## 3.2 Keywords via an alternating wrapper/filter method

The algorithm in Section 3.1 gives a method for $n$-gram selection for fixed $n$. Now, suppose we want to find keywords for arbitrary $n$. One possible hypothesis for keyword selection is that since higher order $n$-grams are discriminative, lower order $n$-grams in the keywords will also be discriminative. Therefore, it makes sense to finding distinguishing lower order $n$-grams and then construct longer units from these. On the basis of this idea, we propose the following algorithm for keyword construction:

**Keyword Building Algorithm**

- Start with an initial value of $n = n_{\mathrm{s}}$. Initialize the set, $\mathcal{S}'_n$, to all possible $n$-grams of phones including lower order grams. By default, let $\mathcal{S}_1$ be the set of all phones.
- (Wrapper Step) General $n$. Apply the feature selection algorithm in Section 3.1 to produce a subset of distinguishing $n$-grams, $\mathcal{S}_n \subset \mathcal{S}'_n$.
- (Filter Step) Construct a new set of $(n+1)$-grams by juxtaposing elements from $\mathcal{S}_n$ with phones. Nominally, we take this step to be juxtaposition on the right and left, $\mathcal{S}'_{n+1} = \{dp, qd | d \in \mathcal{S}_n, p \in \mathcal{S}_1, q \in \mathcal{S}_1\}$.
- Iterate to the wrapper step.
- Output: $\mathcal{S}_n$ at some stopping $n$.

A few items should be noted about the proposed keyword building algorithm. First, we call the second feature selection process a filter step, since induction has not been applied to the $(n+1)$-gram features. Second, note that the purpose of the filter step is to provide a candidate set of possible $(n+1)$-grams which can then be more systematically reduced. Third, several potential algorithms exist for the filter step. In our experiments and in the algorithm description, we nominally append one phone to the beginning and end of an $n$-gram. Another possibility is to try to combine overlapping $n$-grams. For instance, suppose the keyword is `some_people` which has phone transcript `s_ah_m_p_iy_p_l`. Then, if we are looking at $4$-grams, we might see as top features `s_ah_m_p` and `p_iy_p_l` and combine these to produce a new keyword.

## 3.3 Keyword Implementation

The expected $n$-gram counts were computed from lattices using the forward-backward algorithm. Equation (8) gives the posterior probability of a connected sequence of arcs in the lattice where *src_nd*$(a)$ and *dst_nd*$(a)$ are the source and destination node of arc $a$, $\ell(a)$is the likelihood associated with arc $a$, $\alpha(n)$ and $\beta(n)$ are the forward and backward probabilities of reaching node $n$ from the beginning or end of the lattice $\mathcal{L}$ respectively, and $\ell(\mathcal{L})$ is the total likelihood of the lattice (the $\alpha(\cdot)$ of the final node or $\beta(\cdot)$ of the initial node of the lattice).

$$p(a_j, ..., a_{j+n}) = \frac{\alpha(src\_nd(a_j))\ell(a_j)\dots\ell(a_{j+n})\beta(dst\_nd(a_{j+n}))}{\ell(\mathcal{L})} \tag{8}$$

Now if we define the posterior probability of a node $p(n)$ as $p(n) = (\alpha(n)\beta(n))/\ell(\mathcal{L})$. Then equation (8) becomes:

$$p(a_j, ..., a_{j+n}) = \frac{p(a_j)\dots p(a_{j+n})}{p(src\_nd(a_{j+1}))\dots p(src\_nd(a_{j+n}))}. \tag{9}$$

Equation (9) is attractive because it provides a way of computing the path posteriors locally using only the individual arc and node posteriors along the path. We use this computation along with a *trie* structure [15] to compute the posteriors of our keywords.

# 4 Experiments

## 4.1 Language Recognition Experimental Setup

The phone recognizer described in Section 2.1 was used to generate lattices across a train and an evaluation data set. The training data set consists of more than 360 hours of telephone speech

spanning 13 different languages and coming from a variety of different sources including Callhome, Callfriend and Fisher. The evaluation data set is the NIST 2005 Language Recognition Evaluation data consisting of roughly 20,000 utterances (with duration of 30, 10 or 3 seconds depending on the task) coming from three collection sources including Callfriend, Mixer and OHSU. We evaluated our system for the 30 and 10 second task under the the NIST 2005 closed condition which limits the evaluation data to 7 languages (English, Hindi, Japanese, Korean, Mandarin, Spanish and Tamil) coming only from the OHSU data source.

The training and evaluation data was segmented using an automatic speech activity detector and segments smaller than 0.5 seconds were thrown out. We also sub-segmented long audio files in the training data to keep the duration of each utterance to around 5 minutes (a shorter duration would have created too many training instances). Lattice arcs with posterior probabilities lower than $10^{-6}$ were removed and lattice expected counts smaller than $10^{-3}$ were ignored. The top and bottom 600 ranking keywords for each language were selected after each training iteration. The support vector machine was trained using a kernel formulation which requires pre-computing all of the kernel distances between the data points and using an alternate kernel which simply indexes into the resulting distance matrix (this approach becomes difficult when the number of data points is too large).

### 4.2  Language Recognition Results (Qualitative and Quantitative)

To get a sense of how well our keyword building algorithm was working, we looked at the top ranking keywords from the English model only (since our phone recognizer is trained using the English phone set). Table 1 summarizes a few of the more compelling phone 5-grams, and a possible keyword that corresponds to each one. Not suprisingly, we noticed that in the list of top-ranking $n$-grams there were many variations or partial $n$-gram matches to the same keyword, as well as $n$-grams that didn't correspond to any apparent keyword.

The equal error rates for our system on the NIST 2005 language recognition evaluation are summarized in Table 2. The 4-gram system gave a relative improvement of 12% on the 10 second task and 9% on the 30 second task, but despite the compelling keywords produced by the 5-gram system, the performance actually degraded significantly compared to the 3-gram and 4-gram systems.

Table 1: Top ranking keywords for 5-gram SVM for English language recognition model

| phones | Rank | keyword |
|---|---|---|
| SIL_Y_UW_N_OW | 1 | you know |
| !NULL_SIL_Y_EH_AX | 3 | <s> yeah |
| !NULL_SIL_IY_M_TH | 4 | <s> ??? |
| P_IY_P_AX_L | 6 | people |
| R_IY_L_IY_SIL | 7 | really |
| Y_UW_N_OW_OW | 8 | you know (var) |
| T_L_AY_K_SIL | 17 | ? like |
| L_AY_K_K_SIL | 23 | like (var) |
| R_AY_T_SIL_!NULL | 27 | right </s> |
| HH_AE_V_AX_N | 29 | have an |
| !NULL_SIL_W_EH_L | 37 | <s> well |

Table 2: %EER for 10 and 30 second NIST language recognition tasks

| N | 1 | 2 | 3 | 4 | 5 |
|---|---|---|---|---|---|
| 10sec | 25.3 | 16.5 | 11.3 | 10.0 | 13.6 |
| 30sec | 18.3 | 07.4 | 04.3 | 03.9 | 05.6 |

### 4.3 Topic Recognition Experimental Setup

Topic recognition was performed using a subset of the phase I Fisher corpus (English) from LDC. This corpus consists of $5,851$ telephone conversations. Participants were given instructions to discuss a topic for $10$ minutes from $40$ different possible topics. Topics included "Education", "Hobbies," "Foreign Relations", etc. Prompts were used to elicit discussion on the topics. An example prompt is:

> **Movies:** Do each of you enjoy going to the movies in a theater, or would you rather rent a movie and stay home? What was the last movie that you saw? Was it good or bad and why?

For our experiments, we used $2750$ conversation sides for training. We also constructed development and test sets of $1372$ conversation sides each. The training set was used to find keywords and models for topic detection.

### 4.4 Topic Recognition Results

We first looked at top ranking keywords for several topics; some results are shown in Table 3. We can see that many keywords show a strong correspondence with the topic. Also, there are partial keywords which correspond to what appears to be longer keywords, e.g. "eh_t_s_ih_k" corresponds to *get sick*.

As in the language recognition task, we used EER as the performance measure. Results in Table 4 show the performance for several $n$-gram orders. Performance improves going from 3-grams to 4-grams. But, as with the language recognition task, we see a degradation in performance for 5-grams.

## 5 Conclusions and future work

We presented a method for automatic construction of keywords given a discriminative speech classification task. Our method was based upon successively building longer span keywords from shorter span keywords using phones as a fundamental unit. The problem was cast as a feature selection problem, and an alternating filter and wrapper algorithm was proposed. Results showed that reasonable keywords and improved performance could be achieved using this methodology.

Table 3: Top keyword for 5-gram SVM in Topic Recognition

| Topic | Phones | Keyword |
|---|---|---|
| Professional Sports on TV | S_P_AO_R_T | sport |
| Hypothetical: Time Travel | G_OW_B_AE_K | go back |
| Affirmative Action | AX_V_AE_K_CH | [affirmat]**ive act**[ion] |
| US Public Schools | S_K_UW_L_Z | schools |
| Movies | IY_V_IY_D_IY | DVD |
| Hobbies | HH_OH_B_IY_Z | hobbies |
| September 11 | HH_AE_P_AX_N | happen |
| Issues in the Middle East | IH_Z_R_IY_L | Israel |
| Illness | EH_T_S_IH_K | [g]**et sick** |
| Hypothetical: One Million Dollars to leave the US | Y_UW_M_AY_Y | you may |

Table 4: Performance of Topic Detection for Different $n$-gram orders

| $n$-gram order | 3 | 4 | 5 |
|---|---|---|---|
| EER (%) | 10.22 | 8.95 | 9.40 |

Numerous possibilities exist for future work on this task. First, extension and experimentation on other tasks such as dialect and speaker recognition would be interesting. The method has the potential for discovery of new interesting characteristics. Second, comparison of this method with other feature selection methods may be appropriate [16]. A third area for extension is various technical improvements. For instance, we might want to consider more general keyword models where skips are allowed (or more general finite state transducers [17]). Also, alternate methods for the filter for constructing higher order $n$-grams is a good area for exploration.

## Footnotes

*This work was sponsored by the Department of Homeland Security under Air Force Contract FA8721-05-C-0002. Opinions, interpretations, conclusions, and recommendations are those of the authors and are not necessarily endorsed by the United States Government.

## References

[1] W. M. Campbell, J. P. Campbell, D. A. Reynolds, D. A. Jones, and T. R. Leek, "Phonetic speaker recognition with support vector machines," in *Advances in Neural Information Processing Systems 16*, Sebastian Thrun, Lawrence Saul, and Bernhard Schölkopf, Eds. MIT Press, Cambridge, MA, 2003.

[2] W. M. Campbell, T. Gleason, J. Navratil, D. Reynolds, W. Shen, E. Singer, and P. Torres-Carrasquillo, "Advanced language recognition using cepstra and phonotactics: MITLL system performance on the NIST 2005 language recognition evaluation," in *Proc. IEEE Odyssey*, 2006.

[3] Bin Ma and Haizhou Li, "A phonotactic-semantic paradigm for automatic spoken document classification," in *The 28th Annual International ACM SIGIR Conference, Brazil*, 2005.

[4] Lu-Feng Zhai, Man hung Siu, Xi Yang, and Herbert Gish, "Discriminatively trained language models using support vector machines for language identification," in *Proc. IEEE Odyssey: The Speaker and Language Recognition Workshop*, 2006.

[5] T. Joachims, *Learning to Classify Text Using Support Vector Machines*, Kluwer Academic Publishers, 2002.

[6] W. M. Campbell, F. Richardson, and D. A. Reynolds, "Language recognition with word lattices and support vector machines," in *Proceedings of ICASSP*, 2007, pp. IV–989 – IV–992.

[7] Petr Schwarz, Matejka Pavel, and Jan Cernocky, "Hierarchical structures of neural networks for phoneme recognition," in *Proceedings of ICASSP*, 2006, pp. 325–328.

[8] Linguistic Data Consortium, "Switchboard-2 corpora," http://www.ldc.upenn.edu.

[9] "ICSI QuickNet," http://www.icsi.berkeley.edu/Speech/qn.html.

[10] S. Young, Gunnar Evermann, Thomas Hain, D. Kershaw, Gareth Moore, J. Odell, D. Ollason, V. Valtchev, and P. Woodland, *The HTK book*, Entropic, Ltd., Cambridge, UK, 2002.

[11] L. Rabiner and B.-H. Juang, *Fundamentals of Speech Recognition*, Prentice-Hall, 1993.

[12] Ronan Collobert and Samy Bengio, "SVMTorch: Support vector machines for large-scale regression problems," *Journal of Machine Learning Research*, vol. 1, pp. 143–160, 2001.

[13] Avrim L. Blum and Pat Langley, "Selection of relevant features and examples in machine learning," *Artificial Intelligence*, vol. 97, no. 1-2, pp. 245–271, Dec. 1997.

[14] I. Guyon, J. Weston, S. Barnhill, and V. Vapnik, "Gene selection for cancer classification using support vector machines," *Machine Learning*, vol. 46, no. 1-3, pp. 389–422, 2002.

[15] Konrad Rieck and Pavel Laskov, "Language models for detection of unknown attacks in network traffic," *Journal of Computer Virology*, vol. 2, no. 4, pp. 243–256, 2007.

[16] Takaaki Hori, I. Lee Hetherington, Timothy J. Hazen, and James R. Glass, "Open-vocabulary spoken utterance retrieval using confusion neworks," in *Proceedings of ICASSP*, 2007.

[17] C. Cortes, P. Haffner, and M. Mohri, "Rational kernels," in *Advances in Neural Information Processing Systems 15*, S. Thrun S. Becker and K. Obermayer, Eds., Cambridge, MA, 2003, pp. 601–608, MIT Press.

